# Controlled maximal variability along with reliable performance in recurrent neural networks

**Chiara Mastrogiuseppe**
Center for Brain and Cognition, Department of Engineering
Universitat Pompeu Fabra
Barcelona, Spain
`chiara.mastrogiuseppe@upf.edu`

**Rubén Moreno-Bote**
Center for Brain and Cognition, Department of Engineering,
Serra Húnter Fellow Programme
Universitat Pompeu Fabra
Barcelona, Spain
`ruben.moreno@upf.edu`

## Abstract

Natural behaviors, even stereotyped ones, exhibit variability. Despite its role in exploring and learning, the function and neural basis of this variability is still not well understood. Given the coupling between neural activity and behavior, we ask what type of neural variability does not compromise behavioral performance. While previous studies typically curtail variability to allow for high task performance in neural networks, our approach takes the reversed perspective. We investigate how to generate maximal neural variability while at the same time having high network performance. To do so, we extend to neural activity the maximum occupancy principle (MOP) developed for behavior, and refer to this new neural principle as NeuroMOP. NeuroMOP posits that the goal of the nervous system is to maximize future action-state entropy, a reward-free, intrinsic motivation that entails creating all possible activity patterns while avoiding terminal or dangerous ones. We show that this goal can be achieved through a neural network controller that injects currents (actions) into a recurrent neural network of fixed random weights to maximize future cumulative action-state entropy. High activity variability can be induced while adhering to an energy constraint or while avoiding terminal states defined by specific neurons' activities, also in a context-dependent manner. The network solves these tasks by flexibly switching between stochastic and deterministic modes as needed and projecting noise onto a null space. Based on future maximum entropy production, NeuroMOP contributes to a novel theory of neural variability that reconciles stochastic and deterministic behaviors within a single framework.

## 1 Introduction

From opening a door to crossing the street, everyday life hinges on reliably executing actions. Despite that, natural behaviors, including repetitive movements from expert athletes [1, 2, 3, 4], exhibit variability [5, 6, 7]. The mechanisms governing the emergence of this variability from the central and peripheral nervous systems remain unclear. Variability of neural activity in motor cortex [8, 9, 10] and subcortical structures [11, 12] controlling muscle movements, as well as state changes of the effectors

due to, e.g., fatigue [13, 14], might contribute to the observed behavioral variability. However, as behavioral variability is also observed at longer time scales during perception [15, 16], decision making [17, 6, 5, 18] and planning [19], neural fluctuations in most parts of the brain [20, 21, 22] might be involved in the generation of variable behavioral repertoires as a whole.

Several mechanisms have been put forward for the generation of neural variability, including synaptic noise within neural circuits [23, 24] or non-linear network interactions leading to variable activity patterns [25, 26, 27, 28]. These proposed mechanisms are predominantly designed to describe variability during spontaneous activity – in the absence of sensory stimuli – [22, 20] or in response to simple stimuli [29, 30], most frequently outside a complex task. In other theoretical studies where the goal is to maximize network performance in a task, variability is typically suppressed after learning [31, 32, 33]. If anything, the initial noise or activity variability is used as means to regularize and promote exploration during learning [34, 35], but they are considered to be unnecessary thereafter. This approach is also the one taken in state-of-art reinforcement learning, where variability is added during learning, but during task execution policies are forced to be deterministic [36, 37].

Given these two coexisting sides of natural behavior – namely, high task performance in spite of large variability – we ask whether it is possible to have neural networks generating maximal variability while at the same time being able to flexibly switch to deterministic behavioral modes when needed. We surmise that generating neural variability is a fundamental goal of the nervous system, as it enables the exploration of its entire dynamical range. This idea parallels the one that neural activity should be variable to generate the vast behavioral repertoires [38, 8, 39, 40, 41] as the ones empirically observed [5, 18, 42, 11]. The sought neural variability needs to be highly structured in order to avoid non-adaptive behaviors. One relevant example comes from reaching tasks, where neural activity is indeed found to be confined in null spaces as to avoid undesired movements [43]. To address the above question, we build on the maximum occupancy principle (MOP) developed for behavior [44, 45], which posits that agents ought to occupy action-state space by generating all sorts of action-state paths compatible with the dynamical and environmental constraints. Applied to neural activity, we introduce NeuroMOP, which puts forward the hypothesis that the brain should generate maximum entropy in the neural activity paths. Importantly, this entails avoiding the terminal states where further entropy cannot be generated. This principle aligns with a broad body of Reinforcement Learning (RL) literature on reward-free algorithms based on purely-entropic objectives [37, 46, 47]. By optimizing the cumulative sum of future action-state entropy, NeuroMOP emphasizes seeking future variability to the extent that does not compromise performance. By properly defining terminal states as absorbing states where no more entropy can be generated, this principle seeks variability while also generating behaviors that guarantee future 'survival'.

In this paper, we employ random recurrent neural networks (RNNs) of fixed weights as a simplified representation of brain dynamics, and we let them interact with a stochastic input current generator following MOP (Fig. 1). The input current generator (the *agent*) is designed to maximize the entropy – hence, the variability – of the series of currents (a function of the *actions*) it injects in the RNN (the *environment*). As expected, variable currents lead to variable neural activities, but this variability becomes structured in order to avoid dangerous (terminal) states. To bridge neural variability with functionality, we test our architecture in a series of different problems. First, we show that the NeuroMOP network learns to satisfy energy constraints, while generating large neural variability. Second, a subset of RNN neuron activities can be confined within complex regions while remaining free within the region, thereby 'drawing' different symbols, also in a context dependent manner. Crucially, the NeuroMOP network not only learns to effectively solve tasks, but it maintains a high dimensionality of action signals whenever possible, allowing the visitation of a wide range of activity patterns. By flexibly reducing its dimensionality when close to terminal states, we show that low dimensionality is an emergent property under constrained tasks where higher dimensionality is the default mode.

## 2 Methods

### 2.1 NeuroMOP architecture: controller and RNN

The NeuroMOP architecture consists of a controller (*agent*) injecting currents into an RNN of fixed random weights (*environment*) (Fig. 1). The state $x \in \mathbb{R}^N$ of the RNN, where $x_i$ is the activity of

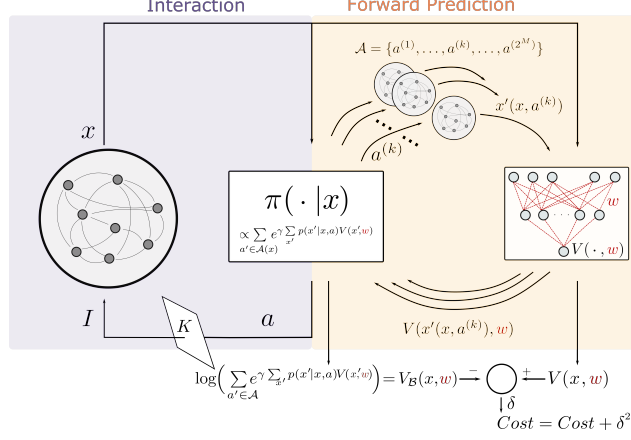

Figure 1: Schematic of the NeuroMOP network. At each time step the controller reads the activity $x$ (state) of the RNN and samples an action $a$ from the policy $\pi(\cdot|x)$. Using the optimal policy in Eq. 4 requires predicting the effect that each of the possible actions $a^{(k)} \in \mathcal{A}(x)$ would have on the state of the RNN by computing the successor state $x'(x, a^{(k)})$ and then evaluating the corresponding value function $V\left(x'(x, a^{(k)})\right)$ in that state. The value function is approximated by a feedforward network (FFN). Once sampled, the low-dimensional action $a$ is expanded and transformed into currents $I$ via a matrix $K$ and fed into the RNN. Next, the RNN state evolves one time step and the loop is repeated. The weights of the FFN are trained via gradient descent using as cost function the Bellman error stored along a batch of trajectories.

neuron $i = 1, \ldots, N$, follows the dynamics

$$x_i(t+1) = x_i(t) + \delta t \left( -\frac{x_i}{\tau} + \Phi \left( \sum_{j=1}^{N} J_{ij} x_j + I_i(t) \right) \right) , \tag{1}$$

where $\Phi(\cdot)$ is a non-linear transfer function, $\delta t$ is the integration time step of the dynamics, and $I_i(t)$ are currents injected by the controller in the RNN neurons. The recurrent connections of the RNN are fixed and sampled from a normal distribution, $J_{ij} \sim \mathcal{N}(0, g^2/N)$. We use a saturating transfer function $\Phi(\cdot) = \tanh(\cdot)$, which leads to chaotic dynamics when the internal recurrent connections are strong enough [25] (see Fig. 2a). Results for RNNs with non-saturating (ReLU) transfer functions are shown in Appendix F.

At time $t$, the controller samples a random *action* $a$ from a state-dependent, stationary policy $\pi(\cdot|x)$. The action is an $M$-dimensional discrete vector $a \in \mathcal{A}(x)$; we will consider below the presence of terminal states $x^\dagger$, where the number of available actions is drastically reduced. Based on the generated action $a$, the controller injects into the $i$-th neuron the current

$$I_i(t) = \rho \sum_{k=1}^{M} K_{ik} a_k , \tag{2}$$

where $K_{ik} \sim U(0, 1)$ are positive input weights sampled from a uniform distribution and $\rho$ is a parameter that scales the strength of the current. Thus, internal actions $a$ are expanded via the random matrix $K$ into $N$-dimensional currents. This expansion allows us to study the harder problem of controlling the RNN's dynamics using actions with reduced dimensionality, i.e., $M \ll N$, but our framework also works for projections, $M > N$.

The controller follows MOP, that is, it aims at occupying action-state path space [44]. Here we restrict ourselves to action paths, and show the general action-state framework in Appendix D. We assume that the controller gets an intrinsic reward of $-\ln \pi(a|x)$ for generating action $a$ when the network is in state $x$ at time $t$, being the largest when the generated action has low probability under the current policy $\pi$. The controller does not greedily maximize this immediate intrinsic reward at every time step. Instead, the policy $\pi$ is chosen to maximize the *value* function, defined as the expected

discounted sum of future intrinsic rewards

$$V_\pi(x) = \mathbb{E}_{\tau \sim \pi, p}\left[ -\sum_{t=0}^{\infty} \gamma^t \ln \pi(a(t)|x(t)) \right] = \mathbb{E}_{\tau \sim \pi, p}\left[ \sum_t \gamma^t \mathcal{H}\left(\mathcal{A}|x(t)\right) \right] , \qquad (3)$$

with discount factor $0 < \gamma < 1$. Note that this expression takes the form of a sum of future action entropies, where $\mathcal{H}(\mathcal{A}|x) = -\sum_{a \in \mathcal{A}(x)} \pi(a|x) \ln \pi(a|x)$. The expectation is over all paths $\tau = (x, a(0), x(1), a(1), \dots)$ with initial condition $x(0) = x$ generated by sampling actions from the policy $\pi$ and following the state transitions probability $p = p(x(t+1)|x(t), a(t))$ defined by Eq. 1. By virtue of maximizing action path entropy, the MOP agent occupies current and future action space as broadly as possible [44, 45]. Discounted cumulative future action entropy stands as the only measure of occupancy that adheres to the intuitive notion that the occupancy over an action path is the sum of the occupancies over any of its subpaths, the so-called additive property [44].

The optimal policy $\pi^*$ ([44], Appendix C) maximizing the value function is

$$\pi^*(a|x) = \frac{1}{Z(x)} e^{\gamma \sum_{x'} p(x'|x,a) V^*(x')} , \qquad (4)$$

where $Z(x) = \sum_{a \in \mathcal{A}(x)} e^{\gamma \sum_{x'} p(x'|x,a) V^*(x')}$ is the partition function and $V^*(x)$ is the optimal value following the optimal policy, defined as

$$V^*(x) = \ln Z(x) = \ln \sum_{a \in \mathcal{A}(x)} e^{\gamma \sum_{x'} p(x'|x,a) V^*(x')} . \qquad (5)$$

In our specific implementation with deterministic dynamics (Eq. 1), the transition probability $p(x'|x, a)$ is a delta function, and so the successor state $x'$ is uniquely determined by the current state $x$ and the action $a$, $x'(x, a)$. Our algorithms work well also for RNNs with noisy dynamics (Appendix E).

We will define different problems by choosing specific state-dependent action sets, so that the available set of actions $a$ depends on $x$, $a \in \mathcal{A}(x)$. Specifically, we define terminal states, denoted $x^\dagger$, as absorbing states the network cannot escape from and where `doing nothing` is the only available action. These terminal states might represent detrimental state regions, e.g., too high neural activity, or other adverse activity patterns resulting in significant external penalties, such as the falling of an agent to the floor. With `doing nothing` being the only action, entering a terminal state is an irreversible process leading to an intrinsic reward of always zero from that point onwards, i.e., $-\ln \pi(a = \texttt{nothing}|x^\dagger) = \ln 1 = 0$, as no further action entropy can be generated. Therefore, by definition, $V_\pi(x^\dagger) = 0$ for any policy. Terminal states can be considered 'dead' states of the network, and they will be naturally avoided to keep maximizing future action path entropy. In our implementations, non-terminal states share the same action set of $M-$dimensional binary actions $a_k \in \{-1, 1\} \ \forall k = 1, \dots, M$. This does not imply that all non-terminal states are equally desirable; via the computation of the value function, the network naturally exhibits less preference for 'bad' states that increase the likelihood of encountering terminal states in the future. We will show that the structure of terminal regions, along with the network dynamics, leads to complex, rich, variable behaviors without the need to specify an extrinsic reward function. In essence, MOP tells agents what *not* to do, and thus it does not restrict behavior. In contrast, standard extrinsic reward maximization tells agents what to do, inevitably limiting behavior. Note that maximizing action path entropy entails striking a balance between maximizing immediate and future entropy, with behaviors that can become *locally* very deterministic if this *globally* opens up larger repertoires of possible action courses, i.e., larger future action entropy.

## 2.2 Value function approximator by minimizing the Bellman error

As our problem involves a high-dimensional continuous state space ($N = 100$), we use a feed-forward network (FFN) to approximate the optimal value function $V^*(x)$ in Eq. 5 with $V(x, w)$. The FFN with parameters $w$ consists of one hidden layer of $N_{hid}$ neurons, an input layer with input the activities $x$ of the RNN, and one single output neuron with activity $V(x, w)$ (Fig. 1).

In order to optimize the parameters of the FFN, we consider $V_\mathcal{B}(x, w)$, the expected evolution of the approximated value function satisfying the Bellman consistency equation, defined as

$$V_\mathcal{B}(x, w) = \ln \left( \sum_{a \in \mathcal{A}(x)} e^{\gamma \sum_{x'} p(x'|x,a) V(x',w)} \right) . \qquad (6)$$

If the approximated value function $V(x, w)$ were equal to the optimal value $V^*(x)$, its expected evolution would coincide with the value function itself, i.e., $V_{\mathcal{B}}(\cdot) = V^*(\cdot)$ (see Eq. 5). Thus, we optimize the weights $w$ of the FFN by minimizing a loss function, defined as the summed squared errors between $V_{\mathcal{B}}(\cdot)$ and $V(\cdot, w)$ over trajectories in each epoch $l = 1, \ldots, N_{ep}$,

$$\mathcal{L}_l(w) = \frac{1}{N_{traj}} \sum_{\tau=1}^{N_{traj}} \frac{1}{t_{end}^{(\tau)}} \sum_{t=1}^{t_{end}^{(\tau)}} \left( V(x^{(\tau)}(t), w) - V_{\mathcal{B}}(x^{(\tau)}(t), w_l) \right)^2 , \tag{7}$$

where the squared error is accumulated over a batch of $N_{traj}$ paths $\tau = \left( x^{(\tau)}(0), x^{(\tau)}(1), \ldots, x^{(\tau)}(t), \ldots \right)$, and the path lifetime $t_{end}^{(\tau)}$ is the minimum between the time when the network reaches a terminal state and the maximum episode duration $T$. Each path $\tau$ is generated by sampling actions from the policy in Eq. 4 with the same initial condition $x(0) = x$. The policy depends on the specific values of the FFN weights at epoch $l$. The parameters of the network are updated at each epoch $l$ using Adam as optimizer.

Our results have proven to be stable also for FFNs only receiving as an input $N_{inp} < N$ activities randomly selected from the $N-$dimensional state, denoted $\tilde{x}$. If needed, the input can be extended to include any required extra-information or $\tilde{x}$ can be constrained to specific neurons. For instance, in the context-dependent constraints problem defined in Sec. 3.2.1, we added extra units to the FFN input layer to flag (via a one-hot vector) the context. Including in $\tilde{x}$ the readout neurons $(x_1, x_2)$ improved stability and performance.

## 2.3 The reward-maximizing network

To provide a comparison for the NeuroMOP network, we introduce the R network, which aims at maximizing the discounted sum of future extrinsic reward. To ensure a fair comparison, we incorporate the notion of survivability present in MOP by assigning to the R network in state $x$ and taking action $a$ the extrinsic reward

$$r(x, a) = \begin{cases} 1 & \text{if } x'(x, a) \neq x^\dagger \\ 0 & \text{if } x'(x, a) = x^\dagger \end{cases}, \tag{8}$$

where $x'(x, a)$ is the state evolution of the RNN as defined in Eq. 1. To allow the generation of variable trajectories also by this network, stochasticity in the action selection is implemented by endowing the network with an $\epsilon$-greedy policy defined as

$$a \sim \pi_\epsilon(\cdot|x) = \begin{cases} \underset{a}{\operatorname{argmax}} \ V_\epsilon(x'(x, a)) & \text{with probability } 1 - \epsilon \\ \text{random} & \text{with probability } \epsilon \end{cases}, \tag{9}$$

where $V_\epsilon(x)$ is the expected future cumulative reward when following this policy, which can be written recursively as

$$V_\epsilon(x) = \mathbb{E}_{a \sim \pi_\epsilon, x' \sim p} \left[ r(x, a) + \gamma V_\epsilon(x') \right] . \tag{10}$$

In this framework $\epsilon$ is a hyperparameter controlling the amount of (random) action variability generated by the network. To better compare the two networks behaviors, the choice of $\epsilon$ is such that the average lifetime of the two networks in each problem is comparable. The value function is approximated using a one-hidden layer FFN as in Sec. 2.2. Analogously to MOP, in order to train the network we generate a batch of $N_{traj}$ paths $\tau$ starting with the same initial conditions $x^{(\tau)}(0) = x$ and minimize the loss function defined as the summed squared error between the approximated value $V_\epsilon(x, w)$ and its expected evolution following the Bellman equation in Eq. 10.

# 3 Results

## 3.1 Energy constraint

To test whether NeuroMOP can produce maximal variability under strict constraints, we first study a scenario where a terminal state is reached when the overall level of the RNN's activities is high. Specifically, $x^\dagger$ are all the states where the energy, defined as a function of $x$, exceeds a certain value $E(x^\dagger) > L$ (see Appendix B). In this way, we implement the idea that high activity is detrimental,

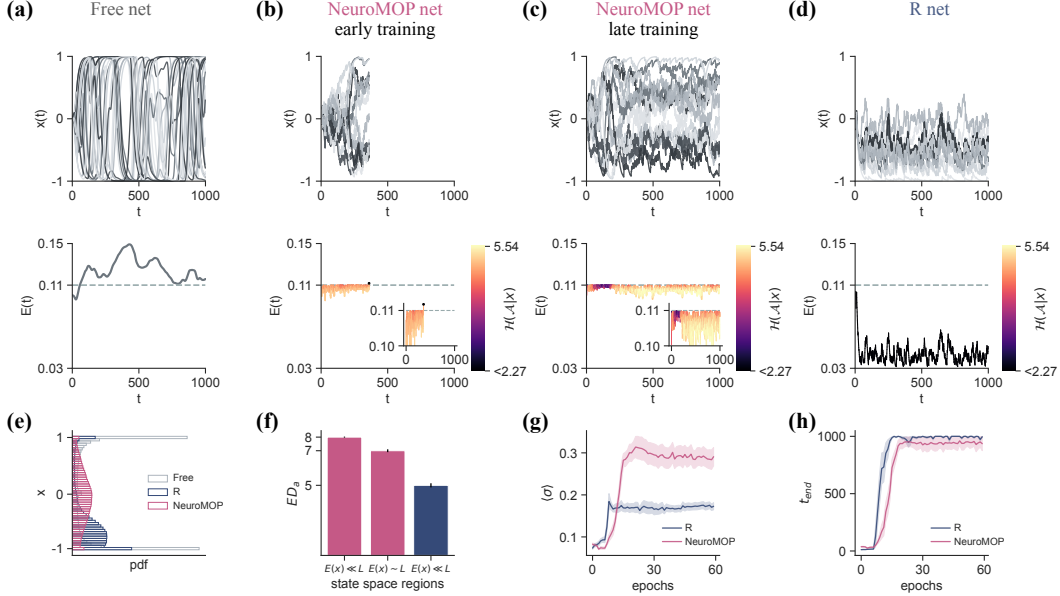

Figure 2: Energy constraint. **(a)** In the free network, the RNN shows chaotic activity (top panel) with high energy consumption (bottom) above threshold (dashed line). In the top panel, each line represents the activity of a randomly sampled neuron of the RNN. A single trial is shown. **(b)** Early in training ($\sim 10$ epochs), the NeuroMOP network quickly reaches a terminal state by crossing the energy threshold (dashed line, bottom panel). Indeed, the large action entropy throughout the trajectory suggests no state-dependent action entropy adjustment. Inset provides a zoom of the energy close to the boundary. **(c)** After training, the NeuroMOP network is able to avoid terminal states for the whole duration of the trajectory ($T = 1000$) by reducing its action entropy whenever closer to the energy threshold. Inset as in (b). **(d)** The R network employs completely different and risk-averse solutions. **(e)** Probability density function of the average state occupancy. **(f)** When far from the energy threshold ($E(x) \ll L$), the NeuroMOP network exhibits maximum effective dimensionality ($ED_a \simeq M = 8$), but loses one degree of freedom ($ED_a \simeq M = 7$) when approaching the threshold corresponding to terminal states ($E(x) \sim L$, i.e. $E(x) \in [L - \delta L, L]$, with $\delta L = 0.001$, arbitrary). The R network only lives far from the threshold injecting mainly inhibitory currents and it exhibits a low effective action dimensionality. **(g,h)** With training, both networks increase the average standard deviation of the individual trajectories $\langle \sigma \rangle$, with MOP displaying larger variability (g). Together, they learn to reach the end of the episode $t_{end} = T = 1000$ (h). Averages over $N_{av} = 10$ networks with batches of $N_{traj} = 10$ trajectories; errors are standard errors of the mean ($SEM$).

either because it is costly [48], or because it leads to neural saturation, impeding sensory encoding [49]. We consider a free network (the RNN without external current, Eq. 1 with $I = 0$) in a chaotic state, where the repetitive saturation of the neurons activities leads to high energy consumption over time (Fig. 2a). The NeuroMOP network learns to control the input current in order to generate maximal input entropy while, at the same time, avoiding terminal states and thus surviving for the whole length of the stimulation (Fig. 2b, early training; Fig. 2c, late training). Importantly, we observe that the NeuroMOP network changes dynamical regimes depending on how far the energy consumption is to threshold (Fig. 2c, bottom panel, inset): when the energy is close to threshold, the action entropy reduces, and therefore the policy becomes more deterministic. In contrast, when the energy is far from threshold, action entropy rises again and the policy increases its stochasticity. Moreover, the policy dimensionality, as measured by the effective dimensionality of the currents (see Appendix A), is lower when close to the threshold compared to further away (Fig. 2f), projecting noise to a lower dimensional manifold such that actions (i.e., currents) that would push the network above threshold are suppressed. Overall, these results show that the NeuroMOP network flexibly changes from a highly stochastic to a more deterministic policy depending on the network state.

Comparing NeuroMOP with the R network, we find that the R network employs different solutions, showcasing a preference for risk-averse solutions (Fig. 2d). As a long lifetime is encouraged by the

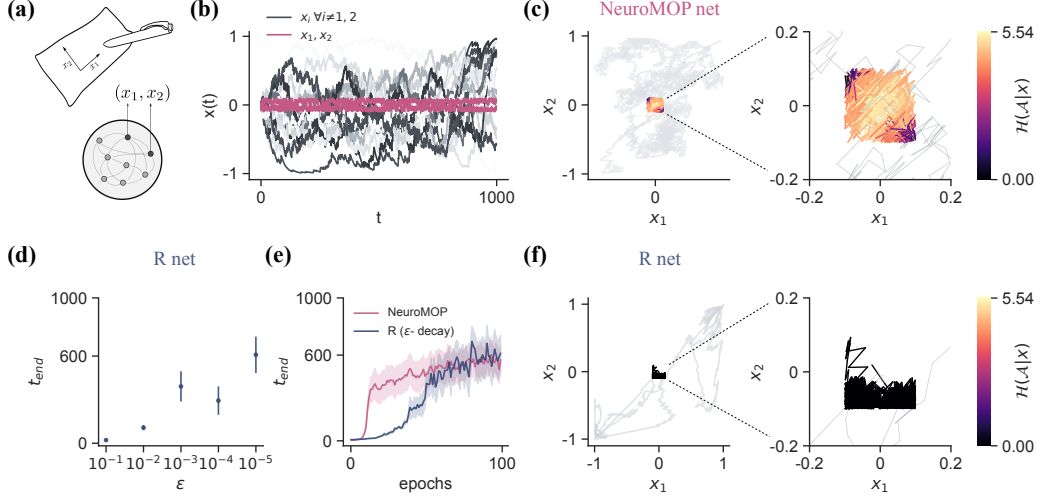

Figure 3: Constrained neural space. **(a)** Terminal states are defined as the boundaries of a square in the activity space of two randomly selected RNN's neurons $(x_1, x_2)$. **(b-c)** As a result, the NeuroMOP network confines the activities $x_1$ and $x_2$ within the square boundaries (panel b, magenta traces), while it 'draws' the square by filling its inside (panel c, colored line, representing one trajectory of the two readout neurons). In contrast, in the space of any other pair of neurons $(x_i, x_j)$ $i, j > 2$ activities spread in space (grey line, representing one trajectory). A zoom-in of the readout space shows that the NeuroMOP network adapts the action entropy based on the state proximity to the boundaries (colorbar, c, right panel). **(d)** The R network, following an $\epsilon - greedy$ policy, fails to avoid the terminal states except for extremely small values of $\epsilon$, effectively reducing its action stochasticity to zero. Lifetime computed after training the network for 100 epochs. **(e)** Matching lifetimes for both NeuroMOP and R network with an $\epsilon-$greedy policy with exponential decay (see Appendix B). The R network learns to satisfy the boundaries constraints after the exponential decay has dropped the randomness of the action selection ($\epsilon$) to zero. **(f)** Same as in (c) for the R network. The R network only 'draws' one side of the square with the two readout neurons $(x_1, x_2)$, while the other neurons, as well receiving the external currents, are driven towards the saturating states. Averages are over $N_{av} = 10$ networks with batches of $N_{traj} = 10$ trajectories. Errors are $SEM$.

extrinsic reward, the R network learns to steer the RNN's energy very far from the terminal state, so that the spontaneous action fluctuations given by the stochasticity of the random policy would not harm the overall performance by keeping it far from threshold. The policy found by the R network consists in injecting mostly inhibitory currents, driving the RNN towards the point of minimum energy and effectively 'silencing' the RNN.

Although both networks are able to avoid the terminal states, they lead to different behaviors and space occupancy: while the R network tends to suppress the RNN's activity, the NeuroMOP network exploits the overall range of activities permitted by the terminal states (Fig. 2e). Again, the NeuroMOP network is able to do so by adapting its action entropy in a state dependent manner. Proximity to the terminal state imposes a constraint on the network's activity. In contrast, the R network operates only far from the threshold and, by showing a clear preference for inhibitory actions, exhibits an effective dimensionality significantly lower than the maximum possible one, i.e., $ED_a < M$ (Fig. 2f). As a consequence of the different action selection strategies, the RNN's neurons within the NeuroMOP network display greater variability than those within the R network, with a larger average standard deviation over the trajectories (Fig. 2 g). The choice of $\epsilon$, representing the level of stochasticity, of the R network is such that the two networks have comparable lifetimes (Fig. 2 h).

## 3.2 Constrained neural space

Terminal states can be arbitrarily imposed on the activities of individual neurons or any subset of them, not only globally as in the previous scenario. Here, we test NeuroMOP in a new problem

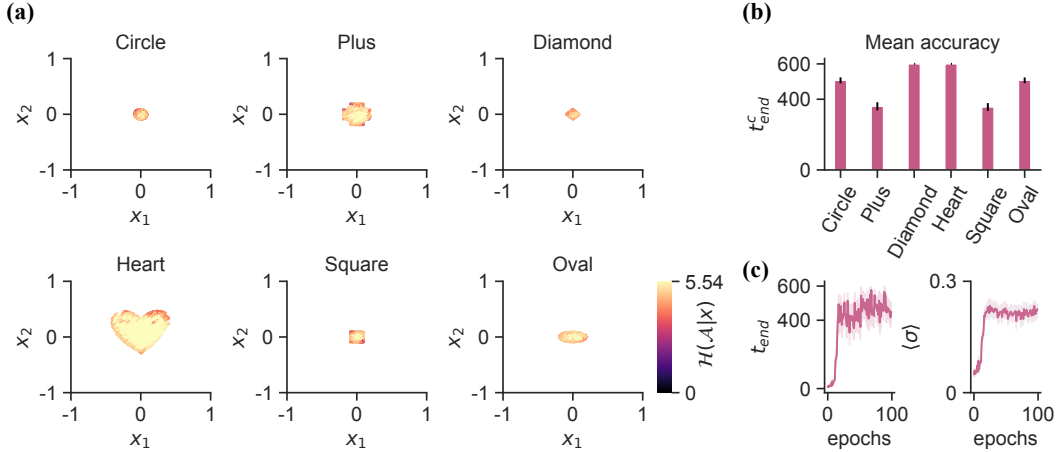

Figure 4: NeuroMOP can constrain a subset of neural activities within different regions of the neural space in a context-dependent manner. The network is informed of the shape it needs to draw via a one-hot vector fed into the value function. **(a)** Example of a single network drawing $C = 6$ different shapes by confining its readout activities $(x_1, x_2)$ within the corresponding activity regions ($T = 5000$). Notably, action entropy is both state and context dependent. One trajectory per context is shown. **(b)** Mean accuracy, measured as the mean lifetime in each context $t_{end}^C$, reflects varying shape difficulty, consistent across networks. **(c)** With training, the NeuroMOP network learns to approach the arbitrary training end of the simulation $t_{end} = T = 600$ (left panel) and to increase the average standard deviation of the individual trajectories $\langle \sigma \rangle$ (right panel). Averages over $N = 10$ networks, with batches of $N_{traj} = 10$ trajectories. Errors are $SEM$.

where terminal states are set directly on two randomly selected readout neurons $(x_1, x_2)$ of the RNN. Specifically, a terminal state is encountered any time $|x_1| > L$ or $|x_2| > L$.

The structure of the terminal states generates interesting behavior in the NeuroMOP network: the network 'draws' a square in the $(x_1, x_2)$ activity space by filling the available area while avoiding the square's boundaries (Fig. 3). In other words, the network occupies all the space allowed by the terminal states after learning. The NeuroMOP network can confine the activities of the two readout neurons even within very small regions of the activity space (Fig. 3b, magenta lines). Notably, as neurons are driven by actions that aim to maximize future cumulative entropy and terminal states are here exclusively set on $x_1$ and $x_2$, all other neurons ($x_i$, $\forall i \neq 1, 2$) occupy a much larger region of the activity space (grey lines, in their own spaces $(x_i, x_j)$). In the $(x_1, x_2)$ space, the network reduces its action entropy when in proximity to terminal states, corresponding to the square boundaries (Fig. 3c). Due to the activity correlations induced by the shared input current, controlling the readout neurons along the anti-diagonal of the square presents challenges for the NeuroMOP network. In those regions, the NeuroMOP network learns the necessity of highly deterministic action selection to avoid terminal states (Fig. 3c, right panel).

While NeuroMOP adapts the stochasticity of its policy to occupy the maximum available space, we find that the R network, following $\epsilon-$greedy policy, fails to do the same for most values of $\epsilon$ (Fig. 3d). The R network's lifetime is comparable to that of the NeuroMOP network for extremely low values of $\epsilon$ (notice decreasing scale), for which the consequent randomness of the action selection is effectively zero. To give more flexibility to the R agent, we allowed the R network to first explore phase space by using an epoch-dependent $\epsilon_l$ with an exponential decay. Starting from a larger $\epsilon_0$ at epoch $l = 0$, and slowly decreasing it, we can match the two lifetimes (Fig. 3e). Despite this, the inherent greediness of the action selection forbids the R network to occupy all the available activity region (Fig. 3f), resulting in a largely repetitive and stereotyped network behavior.

### 3.2.1 Context-dependent neural space constraints

We next wondered about the versatility of the NeuroMOP network to confine neural activity within even more complex boundaries. We introduce context-dependent neural space constraints (see Appendix B), where in each context the set of terminal states in the readout space of two random

activities $(x_1, x_2)$ define different shapes, with different sizes, orientations and border complexity. We augment the input layer of the value approximator described in Sec. 2.2 with $C = 6$ additional nodes, where $C$ represents the number of different shapes where activity has to be confined. Thus, the feedforward value approximator now receives, alongside the network state $x$, a $C$ dimensional one-hot vector indicating the current context.

The same feedforward network, opportunely informed of the context, correctly approximates the value function. Consequently, the NeuroMOP network also adapts its action entropy in a context-dependent manner (Fig. 4a). Notably, while avoiding terminal states, the NeuroMOP network learns to occupy the available state region in each given context. As expected, some terminal states rise greater challenges for the NeuroMOP network to be avoided (Fig. 4b), a feature that is independent on the number of stored contexts $C$. Overall, the NeuroMOP network successfully avoids terminal states while increasing the variability in the network with learning (Fig. 4c).

## 4   Discussion

We have introduced NeuroMOP, a novel theory that puts forward the idea that neural variability arises from the active generation of future neural activity entropy. We have explored this theory by introducing a mechanism for maximizing and controlling variability in a highly-dimensional RNN in the chaotic regime. Contrary to the common idea that excessive variability may impede performance, our model showcases that injecting maximal controlled variability into RNNs actually permits to solve different 'tasks', indirectly defined by the structure of terminal states. By allowing for a diverse array of actions according to the state, this variability enables the network to explore a wider solution space, potentially leading to more effective adaptations [50, 51, 52].

We tested our network in a series of scenarios. First, we introduced an energy threshold on the network activity. Energy constraints may have been likely selected by evolution, as brain activity is costly both during information processing and at rest [53, 48]. By observing that the network always keeps energy consumption close to the threshold without exceeding it, our results align with the idea that sustained, controlled energy consumption could actually be beneficial [54]. In a second series of problems, we showed that the NeuroMOP network can avoid terminal states in the readout space while increasing the variability in the subspaces where no boundaries are set. Therefore, long lifetime is achieved by flexibly switching between deterministic and stochastic dynamical regimes when needed. Additionally, we show that our algorithm is capable of solving problems often tackled through extrinsic rewards, such as balancing a cartpole (Appendix G), or scenarios where more deterministic behavioral modes are required, like traversing a narrow corridor in neural space (Appendix H). In addition, we show that introducing extrinsic rewards in the MOP framework largely reduces the variability of the network behavior (Appendix I).

When comparing MOP with other systems that also generate variability, like a reward-maximizing (R) network with epsilon-greedy noise, we find that R networks can only avoid terminal states after extensive training and only by quenching the source of randomness. We remark that MOP agents face as well the drawback of stochasticity as their policy follows a Boltzmann distribution (Eq. 4), and thus a non-zero probability is assigned to all actions regardless of the state. Despite that, MOP agents overcome this tendency by adapting their randomness via the computation of the value function, which trades-off immediate with future variability. This is in contrast with R agents, where the stochasticity parameter $\epsilon$ is state independent. These results suggest that state adaptation of stochasticity is a relevant property we might expect in intelligent systems.

By keeping the weights of the RNN fixed, we depart from the common practice of training networks. Weights training usually leads to activities exploiting the saturation state of the neurons [35, 33]. We conjecture this dynamics to be unrealistic, as biological neurons largely display activities that are well below their maximum values [55, 56, 49]. Analogously, saturation is undesirable even in artificial neural networks due to the vanishing curvature of the loss landscape. By favoring states and actions with low probability, the NeuroMOP network leads to the more uniform occupation as possible, avoiding saturation and encouraging neurons to stay in a 'healthy' regime, i.e., a regime suitable for computation [57]. As well, we have demonstrated that NeuroMOP can control high-dimensional chaotic RNNs. Future research should investigate how MOP-driven input currents affect the RNN's regime. We anticipate that MOP currents will stabilize neural trajectories, consistent with operating at

the edge of chaos [58]. Characterizing chaoticity as a function of the input properties (e.g., magnitude) [59] is a promising direction.

In our system, an external stochastic input current generator is designed to maximize its cumulative entropy impinging onto the neurons of an RNN. The idea of specialized circuits serving the role of stochastic input generator is not novel. In songbirds, for instance, the LMAN brain region, part of the neural circuit controlling songs' production, has been largely postulated to fulfill the function of injecting variability into the downstream motor pathways [60, 61]. Allegedly, increased variability in the motor neural activity favors behavioral exploration in the songs' production. Remarkably, it has also been shown that, during courtship, adult birds significantly reduce their vocal variability compared to their solitary singing [60, 62]. This switch of behavior from random to more deterministic modes aligns with our hypothesis of the existence of directed variability in the brain.

Finally, NeuroMOP offers several testable predictions regarding the nature of neural variability we should expect in the brain. Firstly, it predicts that neural variability will persist even after extensive training, which aligns with studies reporting large spiking variability even in well-trained non-human primates [21, 29]. Despite this persistence, our model also suggests that neural variability may decrease when terminal states are sufficiently close, as the network is expected to transition into a more deterministic mode to avoid those states [30, 63]. Finally, our model predicts that reward signaling systems in the brain will also signal intrinsic motivation rewards. This is partially supported by recent studies demonstrating that spontaneous movements elicit dopamine release [11]. Further, we postulate that the visitation of all activity states may increase flexibility and help generalization. Consistent with that, certain activity states are observed to be replayed in the absence of any stimulation in the brain, and several mechanisms in RNNs have been proposed for this phenomenon [64, 65]. NeuroMOP predicts the deterministic reactivation of activity patterns and memories that are relevant for generating higher future behavioral entropy, but the more stochastic reactivation of less relevant memories.

**Limitations**  In the proposed framework, we choose not to approximate the policy, but instead to rely on an 'oracle' to provide the best action following the derived exact analytical expression (Fig. 1, $\pi$ box). Our model could be extended to include a neural network to also approximate the policy $\pi$ using actor critic approaches [66]. Therefore, we do not delve in the process of learning the policy itself. Exploring the policy learning process represents a significant direction for future work. Despite relying on the exact policy, the input current selection has still a high computational cost. In order to partially mitigate this, we introduced the random matrix $K$, which transforms low $M$-dimensional binary actions into high $N$-dimensional currents. Via this matrix, we were able to reduce complexity from $\sim \mathcal{O}\left(2^N\right)$ to $\sim \mathcal{O}\left(2^M\right)$, thereby significantly speeding up the computation, without compromising the convergence of the algorithm. The extension of NeuroMOP to more realistic spiking and Poisson-like variability is another major possible direction. Finally, another interesting direction that we have not addressed here is how to learn the structure of terminal states, and how nearby 'bad' states surrounding terminal states can be learnt and used to speed up learning.

**Conclusion**  Our results demonstrate that maximizing cumulative future action entropy while avoiding terminal states leads to interesting behaviors without the need of defining an extrinsic reward function. Our work shows that NeuroMOP networks can flexibly switch between stochastic and deterministic modes as needed to avoid terminal states. These results contribute to a novel theory of neural variability based on future entropy production, reconciling stochastic and deterministic behaviors within a single framework. Our work highlights a significant limitation in classical neuroscience studies, where limited behavioral repertoires are promoted by the task design and experimental trials terminate upon reaching the goal. In ecological settings, in contrast, agents continuously generate interim goals and elicit new behaviors. NeuroMOP offers a powerful model of neural activity underlying natural behavior.

## Acknowledgments and Disclosure of Funding

This work is supported by the Howard Hughes Medical Institute (HHMI, ref 55008742), Ministry of Science and Innovation, State Research Agency, European Union (Project PID2023-146524NB-I00 financed by MCIN/AEI/10.13039/501100011033/ ERDF, EU) and ICREA Academia to R.M.-B. and AGAUR-FI ajuts from Generalitat de Catalunya/ESF (2024 FI-B3 00020) to C.M.

## References

[1] C. Button, M. MacLeod, R. Sanders, and S. Coleman. Examining movement variability in the basketball free-throw action at different skill levels. *Res Q Exerc Sport*, 74(3):257–269, 2003.

[2] S. Miller. Variability in Basketball Shooting: Practical Implications. *ISBS - Conference Proceedings Archive*, 2000.

[3] K. Davids, P. Glazier, D. Araújo, and R. Bartlett. Movement systems as dynamical systems: the functional role of variability and its implications for sports medicine. *Sports Med*, 33(4):245–260, 2003.

[4] R. Bartlett, J. Wheat, and M. Robins. Is movement variability important for sports biomechanists? *Sports Biomech*, 6(2):224–243, 2007.

[5] S. Recanatesi, U. Pereira-Obilinovic, M. Murakami, Z. Mainen, and L. Mazzucato. Metastable attractors explain the variable timing of stable behavioral action sequences. *Neuron*, 110(1):139–153.e9, 2022.

[6] E. Schulz, R. Bhui, B. C. Love, B. Brier, M. T. Todd, and S. J. Gershman. Structured, uncertainty-driven exploration in real-world consumer choice. *Proceedings of the National Academy of Sciences*, 116(28):13903–13908, 2019.

[7] M. J. Kahana, E. V. Aggarwal, and T. D. Phan. The variability puzzle in human memory. *Journal of Experimental Psychology: Learning, Memory, and Cognition*, 44(12):1857–1863, 2018.

[8] M. M. Churchland, A. Afshar, and K. V. Shenoy. A Central Source of Movement Variability. *Neuron*, 52(6):1085–1096, 2006.

[9] R. J. van Beers. Motor learning is optimally tuned to the properties of motor noise. *Neuron*, 63(3):406–417, 2009.

[10] Y. Mandelblat-Cerf, R. Paz, and E. Vaadia. Trial-by-trial variability of single cells in motor cortices is dynamically modified during visuomotor adaption. *J. Neurosci.*, 29(48):15053–15062, 2009.

[11] J. E. Markowitz, W. F. Gillis, M. Jay, J. Wood, R. W. Harris, R. Cieszkowski, R. Scott, D. Brann, D. Koveal, T. Kula, C. Weinreb, M. A. M. Osman, S. R. Pinto, N. Uchida, S. W. Linderman, B. L. Sabatini, and S. R. Datta. Spontaneous behaviour is structured by reinforcement without explicit reward. *Nature*, 614(7946):108–117, 2023.

[12] K. Doya and T. J. Sejnowski. A Novel Reinforcement Model of Birdsong Vocalization Learning. In *Advances in Neural Information Processing Systems*, volume 7, 1994.

[13] F. Cignetti, F. Schena, and A. Rouard. Effects of fatigue on inter-cycle variability in cross-country skiing. *Journal of Biomechanics*, 42(10):1452–1459, 2009.

[14] J. Legg, D. B. Pyne, S. Semple, and N. Ball. Variability of Jump Kinetics Related to Training Load in Elite Female Basketball. *Sports*, 5(4):85, 2017.

[15] F. Attneave. Multistability in perception. *Scientific American*, 225(6):62–71, 1971.

[16] R. Moreno-Bote, J. Rinzel, and N. Rubin. Noise-induced alternations in an attractor network model of perceptual bistability. *Journal of neurophysiology*, 98(3):1125–1139, 2007.

[17] J. Drugowitsch, R. Moreno-Bote, A.K. Churchland, M.N. Shadlen, and A. Pouget. The cost of accumulating evidence in perceptual decision making. *J. Neurosci.*, 32(11):3612–3628, 2012.

[18] F. Cazettes, L. Mazzucato, M. Murakami, J. P. Morais, E. Augusto, A. Renart, and Z. F. Mainen. A reservoir of foraging decision variables in the mouse brain. *Nat Neurosci*, 26(5):840–849, 2023.

[19] D. Campos, F. Bartumeus, V. Méndez, J. S. Andrade, and X. Espadaler. Variability in individual activity bursts improves ant foraging success. *J R Soc Interface*, 13(125):20160856, 2016.

[20] W. R. Softky and C. Koch. The highly irregular firing of cortical cells is inconsistent with temporal integration of random EPSPs. *J. Neurosci.*, 13(1):334–350, 1993.

[21] D. J. Tolhurst, J. A. Movshon, and A. F. Dean. The statistical reliability of signals in single neurons in cat and monkey visual cortex. *Vision Res*, 23(8):775–785, 1983.

[22] A. Arieli, A. Sterkin, A. Grinvald, and A. Aertsen. Dynamics of ongoing activity: explanation of the large variability in evoked cortical responses. *Science*, 273(5283):1868–1871, 1996.

[23] A. A. Faisal, L. P. J. Selen, and D. M. Wolpert. Noise in the nervous system. *Nat Rev Neurosci*, 9(4):292–303, 2008.

[24] R. Moreno-Bote. Poisson-like spiking in circuits with probabilistic synapses. *PLoS Comput Biol*, 10(7):e1003522, 2014.

[25] H. Sompolinsky, A. Crisanti, and H. J. Sommers. Chaos in Random Neural Networks. *Phys. Rev. Lett.*, 61(3):259–262, 1988.

[26] C. van Vreeswijk and H. Sompolinsky. Chaos in neuronal networks with balanced excitatory and inhibitory activity. *Science*, 274(5293):1724–1726, 1996.

[27] A. Renart, R. Moreno-Bote, X.-J. Wang, and N. Parga. Mean-driven and fluctuation-driven persistent activity in recurrent networks. *Neural Comput*, 19(1):1–46, 2007.

[28] G. Orbán, P. Berkes, J. Fiser, and M. Lengyel. Neural Variability and Sampling-Based Probabilistic Representations in the Visual Cortex. *Neuron*, 92(2):530–543, 2016.

[29] E. Zohary, M. N. Shadlen, and W. T. Newsome. Correlated neuronal discharge rate and its implications for psychophysical performance. *Nature*, 370(6485):140–143, 1994.

[30] M. M. Churchland, B. M. Yu, J. P. Cunningham, L. P. Sugrue, M. R. Cohen, G. S. Corrado, W. T. Newsome, A. M. Clark, P. Hosseini, B. B. Scott, D. C. Bradley, M. A. Smith, A. Kohn, J. A. Movshon, K. M. Armstrong, T. Moore, S. W. Chang, L. H. Snyder, S. G. Lisberger, N. J. Priebe, I. M Finn, D. Ferster, S. I. Ryu, G. Santhanam, M. Sahani, and K. V. Shenoy. Stimulus onset quenches neural variability: a widespread cortical phenomenon. *Nat Neurosci*, 13(3):369–378, 2010.

[31] L. Mazzucato, A. Fontanini, and G. La Camera. Dynamics of multistable states during ongoing and evoked cortical activity. *J. Neurosci.*, 35(21):8214–8231, 2015.

[32] A. Litwin-Kumar and B. Doiron. Slow dynamics and high variability in balanced cortical networks with clustered connections. *Nat Neurosci*, 15(11):1498–1505, 2012.

[33] R. Laje and D. V. Buonomano. Robust timing and motor patterns by taming chaos in recurrent neural networks. *Nat Neurosci*, 16(7):925–933, 2013.

[34] H. Jaeger and H. Haas. Harnessing nonlinearity: predicting chaotic systems and saving energy in wireless communication. *Science*, 304(5667):78–80, 2004.

[35] D. Sussillo and L. F. Abbott. Generating Coherent Patterns of Activity from Chaotic Neural Networks. *Neuron*, 63(4):544–557, 2009.

[36] T. Haarnoja, A. Zhou, P. Abbeel, and S. Levine. Soft Actor-Critic: Off-Policy Maximum Entropy Deep Reinforcement Learning with a Stochastic Actor. In *Proceedings of the 35th International Conference on Machine Learning*, pages 1861–1870, 2018.

[37] E. Hazan, S. Kakade, K. Singh, and A. Van Soest. Provably efficient maximum entropy exploration. In *International Conference on Machine Learning*, pages 2681–2691. PMLR, 2019.

[38] A. Renart and C. K. Machens. Variability in neural activity and behavior. *Curr Opin Neurobiol*, 25:211–220, 2014.

[39] S. Haar, O. Donchin, and I. Dinstein. Individual Movement Variability Magnitudes Are Explained by Cortical Neural Variability. *J. Neurosci.*, 37(37):9076–9085, 2017.

[40] L. Waschke, N. A. Kloosterman, J. Obleser, and D. D. Garrett. Behavior needs neural variability. *Neuron*, 109(5):751–766, 2021.

[41] R. Darshan, W. E. Wood, S. Peters, A. Leblois, and D. Hansel. A canonical neural mechanism for behavioral variability. *Nat Commun*, 8(1):15415, 2017.

[42] P. Dagenais, S. Hensman, V. Haechler, and M. C. Milinkovitch. Elephants evolved strategies reducing the biomechanical complexity of their trunk. *Curr Biol*, 31(21):4727–4737.e4, 2021.

[43] M. T. Kaufman, M. M. Churchland, S. I. Ryu, and K. V. Shenoy. Cortical activity in the null space: permitting preparation without movement. *Nat Neurosci*, 17(3):440–448, 2014.

[44] J. Ramírez-Ruiz, D. Grytskyy, C. Mastrogiuseppe, Y. Habib, and R. Moreno-Bote. Complex behavior from intrinsic motivation to occupy future action-state path space. *Nat Commun*, 15(1):6368, 2024.

[45] R. Moreno-Bote and J. Ramirez-Ruiz. Empowerment, free energy principle and maximum occupancy principle compared. In *NeurIPS 2023 workshop: Information-Theoretic Principles in Cognitive Systems*, 2023.

[46] B. Eysenbach, A. Gupta, J. Ibarz, and S. Levine. Diversity Is All You Need: Learning Skills Without A Reward Function. In *International Conference on Learning Representations*, 2019.

[47] S. Mohamed and D. Jimenez Rezende. Variational Information Maximisation for Intrinsically Motivated Reinforcement Learning. In *Advances in Neural Information Processing Systems*, volume 28, 2015.

[48] J. E. Niven and S. B. Laughlin. Energy limitation as a selective pressure on the evolution of sensory systems. *Journal of Experimental Biology*, 211(11):1792–1804, 2008.

[49] S. Laughlin. A Simple Coding Procedure Enhances a Neuron's Information Capacity. *Zeitschrift für Naturforschung C*, 36(9-10):910–912, 1981.

[50] H. G. Wu, Y. R. Miyamoto, L. N. G. Castro, B. P. Ölveczky, and M. A. Smith. Temporal structure of motor variability is dynamically regulated and predicts motor learning ability. *Nat Neurosci*, 17(2):312–321, 2014.

[51] S. Kirkpatrick, C. D. Jr. Gelatt, and M. P. Vecchi. Optimization by Simulated Annealing. *Science*, 220(4597), 1983.

[52] S. Fusi. Hebbian spike-driven synaptic plasticity for learning patterns of mean firing rates. *Biological Cybernetics*, 87(5-6), 2002.

[53] D. Attwell and S. B. Laughlin. An energy budget for signaling in the grey matter of the brain. *J Cereb Blood Flow Metab*, 21(10):1133–1145, 2001.

[54] C. Chintaluri and T. P. Vogels. Metabolically regulated spiking could serve neuronal energy homeostasis and protect from reactive oxygen species. *Proceedings of the National Academy of Sciences*, 120(48):e2306525120, 2023.

[55] I. Ohzawa, G. Sclar, and R. D. Freeman. Contrast gain control in the cat's visual system. *J Neurophysiol*, 54(3):651–667, 1985.

[56] G. Sclar, P. Lennie, and D. D. DePriest. Contrast adaptation in striate cortex of macaque. *Vision Research*, 29(7):747–755, 1989.

[57] A. Lazar, G. Pipa, and J. Triesch. SORN: a self-organizing recurrent neural network. *Frontiers in Computational Neuroscience*, 3, 2009.

[58] M. Cencini, F. Cecconi, and A. Vulpiani. *Chaos: From Simple Models to Complex Systems*. Series on advances in statistical mechanics. World Scientific, 2010.

[59] S. Takasu and T. Aoyagi. Suppression of chaos in a partially driven recurrent neural network. *Phys. Rev. Res.*, 6:013172, 2024.

[60] M. H. Kao, A. J. Doupe, and M. S. Brainard. Contributions of an avian basal ganglia–forebrain circuit to real-time modulation of song. *Nature*, 433(7026):638–643, 2005.

[61] B. P. Ölveczky, T. M. Otchy, J. H. Goldberg, D. Aronov, and M. S. Fee. Changes in the neural control of a complex motor sequence during learning. *Journal of Neurophysiology*, 106(1):386–397, 2011.

[62] S. C. Woolley and A. J. Doupe. Social Context–Induced Song Variation Affects Female Behavior and Gene Expression. *PLoS Biol*, 6(3):e62, 2008.

[63] R. Nogueira, S. Lawrie, and R. Moreno-Bote. Neuronal Variability as a Proxy for Network State. *Trends in Neurosciences*, 41(4):170–173, 2018.

[64] T. Asabuki and T. Fukai. Predictive learning rules generate a cortical-like replay of probabilistic sensory experiences. *eLife*, (13):RP92712, August 2024.

[65] N. H. Krishna, C. Bredenberg, D. Levenstein, B. A. Richards, and G. Lajoie. Sufficient conditions for offline reactivation in recurrent neural networks. In *UniReps: the First Workshop on Unifying Representations in Neural Models*, 2023.

[66] V. Konda and J. Tsitsiklis. Actor-Critic Algorithms. In *Advances in Neural Information Processing Systems*, volume 12, 1999.

[67] K. Rajan, L. F. Abbott, and H. Sompolinsky. Stimulus-dependent suppression of chaos in recurrent neural networks. *Phys. Rev. E*, 82:011903, 2010.

[68] M. Farrell, S. Recanatesi, G. Lajoie, and E. Shea-Brown. Recurrent neural networks learn robust representations by dynamically balancing compression and expansion. In *NeurIPS 2019 workshop: Real Neurons & Hidden Units: Future directions at the intersection of neuroscience and artificial intelligence*, 2019.

[69] G. Brockman, V. Cheung, L. Pettersson, J. Schneider, J. Schulman, J. Tang, and W. Zaremba. Openai gym. 2016. arXiv:1606.01540.

[70] R. V. Florian. Correct equations for the dynamics of the cart-pole system. In *Center for Cognitive and Neural Studies (Coneural)*, 2005.

# Appendix

## A  Additional Methods

**Effective dimensionality**   We model an external controller sampling actions $a$ living in an $M-$dimensional space. In the NeuroMOP network, the controller aims at occupying action-state path space, and therefore we expect the sampled actions to maximally occupy the action space. To quantify the effective occupation of action space, we introduce the effective dimensionality [67, 68] of the actions as

$$ED_a = \frac{\left(\sum_i \lambda_i\right)^2}{\left(\sum_i \lambda_i^2\right)} \ , \tag{11}$$

where $\lambda_i$ ($i = 1, \ldots, N$) are the eigenvalues of the covariance matrix of the sampled actions. Intuitively, the effective dimensionality quantifies the number of dimensions needed to explain the observed variance in the sampled actions by identifying the dimensions of spread of the signal. In the absence of constraints, the NeuroMOP network would maximally occupy space, uniformly in all the directions of the action space, leading to comparable eigenvalues $\lambda_i \ \forall i = 1, \ldots, M$ and an effective dimensionality close to the full dimensionality of the action space itself ($ED_a \simeq M$). The presence of constraints introduce directions of actions that will be avoided by optimal networks. Along these constrained directions, the sampled actions variances, hence the corresponding eigenvalues, are significantly reduced, resulting in a lower effective dimensionality, i.e. $ED_a < M$. Since actions are sampled from a state-dependent stationary distribution, the effective action dimensionality may vary according to the state. To investigate that, we quantify the effective dimensionality as a function of different states by restricting the covariance matrix, from which the eigenvalues in Eq. 11, to the actions sampled in specific regions of the state space.

**Average standard deviation**   We quantify the induced variability in the RNN's activities by measuring the fluctuations of individual neurons. For this reason, we introduce the average standard deviation as

$$\langle \sigma \rangle = \frac{1}{N_{traj}} \frac{1}{N} \sum_\tau \sum_{i=1}^{N} \sigma_i^{(\tau)} \ , \tag{12}$$

where $\sigma_i^{(\tau)}$ is the standard deviation of the activities of neuron $i \ \forall i = 1, \ldots, N$ along a trajectory $\tau$.

**Parameters of the simulation**   We simulate RNNs in the chaotic state [25, 35] with the parameters reported in Table 1, unless otherwise specified. The parameters defining the algorithm, including the specifics of the feedfoward network and the details of the optimization, are reported in Table 2.

## B  Terminal states

**Energy constraint**   Terminal states are reached whenever the RNN's current energy expenditure exceeds $L = 0.11$ (arbitrary units). Energy is defined as $E(x) = \frac{1}{N} \sqrt{\sum_{i=1}^{N} (x_i + 1)^2}$, the euclidean norm of the activity, translated so that the lowest activity state $x_i = -1 \ \forall i = 1, \ldots, N$ has zero energy. Thus, terminal states $x^\dagger$ are all the states where $E(x^\dagger) > L$. Analogous results have been

Table 1: Hyperparameters for the RNN.

| Parameter | Value |
| --- | --- |
| N | 100 |
| nonlinearity $\Phi(\cdot)$ | tanh, ReLU |
| $\delta$t | 0.05 |
| $\tau$ | 1.0 |
| g | 5.0 |
| $\rho_{tanh}$ | 2.0 |
| $\rho_{ReLU}$ | 5.0 |

Table 2: Hyperparameters for the algorithm, including the parameters of the FFN.

| Parameter | Value |
|---|---|
| action dimensionality $M$ | 8 |
| discount factor $\gamma$ | 0.9 |
| number of hidden layers (FFN) | 1 |
| hidden units per layer | 256 |
| input units $N_{inp}$ | 20 |
| FFN nonlinearity | ReLU |
| $x^\dagger$ | see Sec. Terminal states |
| training epochs $N_{ep}$ | see Sec. Terminal states |
| number of agents $N_{ag}$ | 10 |
| trajectories per batch $N_{traj}$ | 10 |
| optimizer | Adam |
| learning rate $\eta$ | 0.01 |

obtained with different definitions of energy function. Reward-maximizing (R) networks follow an $\epsilon-$greedy policy with $\epsilon = 0.3$. Both networks are trained for $N_{ep} = 60$ epochs.

**Constrained neural space**  Terminal states are all states where $|x_1| > L$ or $|x_2| > L$, with $(x_1, x_2)$ being two arbitrary neurons of the RNN and $L = 0.1$. The R networks follow an $\epsilon-$greedy policy with exponential discount, i.e. at epoch $l$ the probability of sampling a random action decreases as $\epsilon_l = \epsilon_{l-1} \times \delta$, with $\epsilon_0 = 0.5$ and $\delta = 0.9$. The networks are trained for $N_{ep} = 100$ epochs.

**Context-dependent neural space constraints**  Taking two length scales $L_+ = 0.2$ and $L_- = 0.1$, we define terminal states in the different contexts as below. In each epoch, one of the $C$ possible contexts is randomly sampled, and kept fixed for all the trajectories in the epoch. More stable yet slower convergence is obtained if different contexts are sampled in each trajectory of the batch. The NeuroMOP network is trained for $N_{ep} = 100$ epochs. Terminal states in each context are reached whenever the following conditions are met:

**square context**: $|x_1| > L_-$ or $|x_2| > L_-$;

**plus context**: $|x_1| > L_+$ or $|x_2| > L_+$ or $(|x_1| > L_-$ and $|x_2| > L_-)$;

**circle context**: $\sqrt{x_1^2 + x_2^2} > L_+$;

**diamond context**: $|x_1| + |x_2| > L_+$;

**oval context**: $\left(\frac{x_1}{L_+}\right)^2 + \left(\frac{x_2}{L_-}\right)^2 > 1$;

**heart context**: $\left(x_1^2 + x_2^2 - L_-\right)^3 - x_1^2 x_2^3 > 0$.

## C  Optimal policy and value

We show here the derivation of the analytical expression for the optimal policy $\pi^*(a|x)$ in the case of an agent following MOP and maximizing the action-state path entropy [44]. Then, we use this analytical solution to derive the Bellman consistency equation. While in the main text we focused on agents maximizing the action space occupancy, here we take the more general formulation considering both the action and state space occupancy maximization. We include the state entropy in our NeuroMOP network using a small noise approximation in Appendix D.

The MOP agent gets an intrinsic reward over a path equal to $R(\tau) = -\sum_t \gamma^t \ln\left(\pi^\alpha\left(a(t)|x(t)\right) p^\beta\left(x(t+1)|x(t), a(t)\right)\right)$, with discount factor $0 < \gamma < 1$, action $\alpha > 0$ and state $\beta \geq 0$ weights, and where $\tau = (x(0), a(0), x(1), a(1), \ldots, x(t), a(t), \ldots)$ denotes a path of states and actions. Although we use two parameters, $\alpha$ and $\beta$, effectively the number of parameter is only one, their ratio, which measures the relative strength of state over action entropy. Note that the action path entropy maximizer agent is recovered by taking $\beta = 0$ and $\alpha = 1$. The objective of the agent is to maximize the value function $V(x)$, defined as the expected return of the

intrinsic reward as $V(x) = \mathbb{E}_{\tau \sim \pi, p}\left[R(\tau)|x(0) = x\right]$. The Bellman recursive equation enables us to write the value function as the sum of the intrinsic reward the agent receives in the state $x$ and the expected discounted sum of future intrinsic rewards, i.e., the value function in the next state, taking the form

$$
\begin{aligned}
V(x) &= \mathbb{E}_{\tau \sim \pi, p}\left[R(\tau)|x(0) = x\right] \\
&= \mathbb{E}_{\tau \sim \pi, p}\left[-\sum_{t=0}\gamma^t \ln\left(\pi^\alpha\left(a(t)|x(t)\right)p^\beta\left(x(t+1)|x(t), a(t)\right)\right)|x(0) = x\right] \\
&= -\sum_a\sum_{x'}\pi(a|x)p(x'|x,a)\sum_{t=0}\gamma^t \ln\left(\pi^\alpha(a(t)|x(t))p^\beta\left(x(t+1)|x(t), a(t)\right)\right)\Big|_{x(0)=x} \\
&= -\sum_a\sum_{x'}\pi(a|x)p(x'|x,a)\bigg[\ln\left(\pi^\alpha(a|x)p^\beta\left(x'|x,a\right)\right) + \\
&\qquad + \gamma\sum_{t=0}\gamma^t \ln\left(\pi^\alpha(a(t)|x(t))p^\beta\left(x(t+1)|x(t), a(t)\right)\right)\Big|_{x(0)=x'}\bigg] \\
&= -\sum_a\pi(a|x)\alpha\ln\pi(a|x) - \sum_a\pi(a|x)\sum_{x'}p(x'|x,a)\beta\ln p\left(x'|x,a\right) + \\
&\qquad + \gamma\mathbb{E}\left[R(\tau')|x(0) = x'\right] \\
&= \alpha\mathcal{H}(\mathcal{A}|x) + \beta\sum_a\pi(a|x)\mathcal{H}(\mathcal{S}|x,a) + \gamma\sum_a\pi(a|x)\sum_{x'}p(x'|x,a)V(x'),
\end{aligned}
$$

where we recognize the entropy over the action space $\mathcal{H}(\mathcal{A}|x) = -\sum_a\pi(a|x)\ln\pi(a|x)$ and state space $\mathcal{H}(\mathcal{S}|x,a) = -\sum_{x'}p(x'|x,a)\ln p\left(x'|x,a\right)$.

The optimal policy $\pi^*(\cdot|\cdot)$ is the policy maximizing this value function. Therefore, we look for the critical policies $\pi^c(\cdot|\cdot)$ under the constraints $\pi(\cdot|s) \geq 0$ and $\sum_a\pi(a|x) = 1$. Finding the critical points of the Lagrangian function defined as $\mathcal{L} = V(x) - \lambda\left(\sum_a\pi(a|x) - 1\right)$ involves solving

$$
\frac{\partial \mathcal{L}}{\partial \pi(a|x)}\Big|_{\pi=\pi^c} = \frac{\partial V(x)}{\partial \pi(a|x)}\Big|_{\pi=\pi^c} - \lambda(x,x) = 0, \tag{13}
$$

where $\lambda(x,x)$ indicates that the derivatives and the policy are computed in the same state $x$. Deriving the value function with respect to $\pi(a|x)$ gives the desired Lagrange multiplier $\lambda$. By writing $V(x)$ using the Bellman recursive equation we get that

$$
\begin{aligned}
\lambda(x,x) &= \frac{\partial V(x)}{\partial \pi(a|x)}\Big|_{\pi=\pi^c} \\
&= \frac{\partial}{\partial \pi(a|x)}\left[-\sum_{a'}\pi(a'|x)\left(\alpha\ln\pi(a'|x) + \sum_{x'}p(x'|x,a')\left(\beta\ln p(x'|x,a') - \right.\right.\right. \\
&\qquad\left.\left.\left. - \gamma V(x'))\right)\right]\right|_{\pi=\pi^c} \\
&= -\alpha\ln\pi^c(a|x) + \sum_{x'}p(x'|x,a)\left(-\beta\ln p(x'|x,a) + \gamma V_{\pi^c}(x')\right) - \\
&\qquad - \alpha + \gamma\sum_{a'}\pi^c(a'|x)\sum_{x'}p(x'|x,a')\lambda(x',x) \\
&= -\alpha\ln\pi^c(a|x) - \beta\sum_{x'}p(x'|x,a)\ln p(x'|x,a) + \gamma\sum_{x'}p(x'|x,a)V_{\pi^c}(x') + h \tag{14}
\end{aligned}
$$

where we collected in $h = h(x)$ all the terms that are not dependent on $a$. Introducing the partition function $Z(x) = \exp\left(\frac{1}{\alpha}\left(\lambda(x,x) - h(x)\right)\right) = \sum_{a\in\mathcal{A}(x)}\exp\left(\frac{1}{\alpha}\left(\beta\mathcal{H}(\mathcal{S}|x,a) + \gamma\sum_{x'}p(x'|x,a)V_{\pi^c}(x')\right)\right)$ as the normalization constant, the critical policy takes the expression

$$
\pi^c(a|x) = Z(x)^{-1}\exp\left(\frac{1}{\alpha}\left(\beta\mathcal{H}(\mathcal{S}|x,a) + \gamma\sum_{x'}p(x'|x,a)V_{\pi^c}(x')\right)\right). \tag{15}
$$

The corresponding critical expected return is obtained by substituting the critical policy in the Bellman recursive equation as

$$
\begin{aligned}
V^c(x) &= \sum_a \pi^c(a|x) \sum_{x'} p(x'|x,a) \left[ -\alpha \ln \pi^c(a|x) - \beta \ln p(x'|x,a) \right] + \gamma \mathbb{E}[V^c(x')] \\
&= \sum_a \pi^c(a|x) \sum_{x'} p(x'|x,a) \left[ \alpha \ln Z(x) - \beta \ln p(x'|x,a) - \right. \\
&\quad \left. - \alpha \ln \left( e^{\frac{1}{\alpha}\left(\beta \mathcal{H}(\mathcal{S}|x,a) + \gamma \sum_{x'} p(x'|x,a) V^c(x'))\right)} \right) \right] + \gamma \mathbb{E}_{\pi^c}\left[V^c(x')\right] \\
&= \alpha \ln Z(x) - \sum_a \pi^c(a|x) \left( \beta \mathcal{H}(\mathcal{S}|x,a) + \gamma \sum_{x'} p(x'|x,a) V^c(x') \right) + \\
&\quad + \sum_a \pi^c(a|x)\beta \mathcal{H}(\mathcal{S}|x,a) + \gamma \mathbb{E}_{\pi^c}\left[V^c(x')\right] \\
&= \alpha \ln Z(x) = \alpha \ln \sum_{a \in \mathcal{A}(x)} e^{\left(\frac{1}{\alpha}\left(\beta \mathcal{H}(\mathcal{S}|x,a) + \gamma \sum_{x'} p(x'|x,a) V_{\pi^c}(x'))\right)\right)} ,
\end{aligned}
\tag{16}
$$

where we see the expectation value over future states simplifies in the last step.

We now prove that this stationary point corresponds to a maximum of the value function. For this, first note that the value function is continuous and has continuous derivatives with respect to the policy, and therefore the maximum lies either on the boundaries of the policy constraints or it is indeed the critical value. Given a state $x$, the policy boundaries are the points where an (initially available) action $\tilde{a}$ is unavailable, i.e. $\pi(\tilde{a}|x) = 0$. Thus, computing the critical value of the expected return along a boundary leads to the same solution defined in Eq.(16) but the unavailable action $\tilde{a}$ does not appear in the sum over all the possible $a$. As the expected return is an increasing function with the elements making up the sum, the critical value $V^c(x)$ is greater than the expected return along the policy boundaries. The critical value function is therefore the optimal value function, i.e., $V^c(x) = V^*(x)$, and the critical policy is indeed the optimal policy, i.e., $\pi^c(a|x) = \pi^*(a|x)$. Uniqueness of the critical value comes from concavity of the value function.

As discussed in Sec. 2.2, we deal only with approximations of the optimal value function, and consequently our $V(x,w)$ will not exactly satisfy the Bellman consistency equation. We extend the evolution via the Bellman operator defined in Eq. 6 to the case of the action-state occupancy maximization by defining $V_{\mathcal{B}}(x,w)$ as

$$
V_{\mathcal{B}}(x,w) = \alpha \ln Z_{V_{\mathcal{B}}}(x,w) = \ln \sum_a \exp\left( \frac{1}{\alpha}\left( \beta \mathcal{H}(\mathcal{S}|x,a) + \gamma \sum_{x'} p(x'|x,a) V(x',w) \right) \right) .
\tag{17}
$$

As the Bellman consistency equation is satisfied by the optimal value $V^*(x)$, we take the best representation of the weights $w$ of the value approximator to be the one that minimizes the difference between the value $V(x,w)$ and its evolution through the Bellman operator $V_{\mathcal{B}}(x,w)$ by minimizing the loss function defined in Eq. 7.

## D  Approximation of the state entropy term

We introduce here an approximation for the state entropy term in NeuroMOP networks that maximize cumulative future action-*state* entropy. The state entropy term could bring a fundamental contribution in the overall desired large occupancy of the state space when the magnitude of the action signal is weaker. To quantify the state entropy term, we leverage the non-linear dynamics of the RNN. By taking the small noise limit, we assume that a network maximizing state entropy would exhibit a preference for the regions where the non linear dynamics induces larger changes in phase-space volumes, resulting in a larger occupation in the state space. We rewrite the RNN's dynamics in Eq. 1 in differential form as $dx' = dx + \delta t \nabla f_\pi(x,a) dx$ , where $f_\pi(x,a) = -\frac{x}{\tau} + \Phi(Jx + I(a))$ is the dynamics of the RNN for a fixed policy $\pi(\cdot|x)$ and where we made explicit the dependence from the MOP actions. We quantify the changes in the occupation by looking at the changes in the volume in

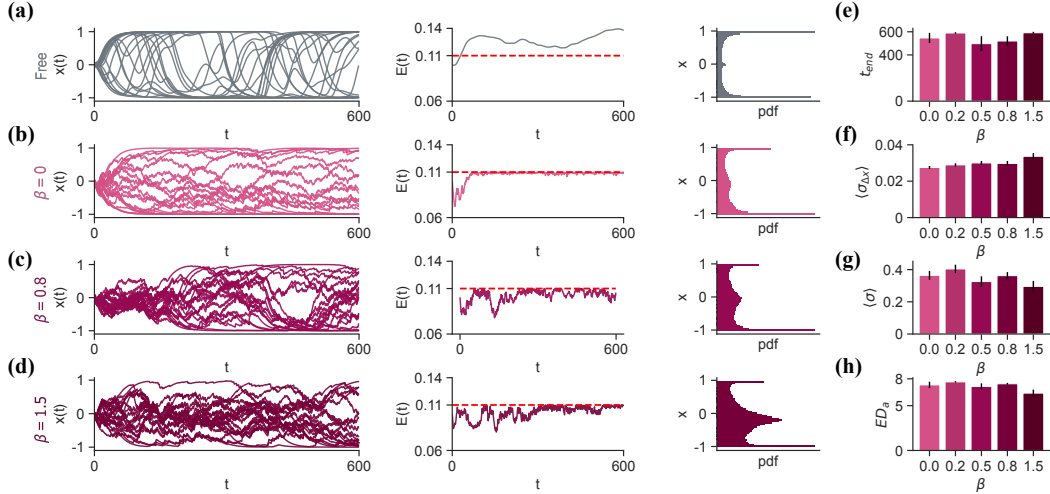

Figure 5: Effect of introducing a 'reward' term approximating the state-entropy. **(a)** The free network is characterized by high energy consumption and the exploitation of the saturating states, with the RNN's neurons alternating between activities $-1$ and $1$. **(b-d)** Same as in (a), but for increasing values of the $\beta \in \{0.0, 0.8, 1.5\}$. With increasing $\beta$, there is a larger average occupancy of the non-saturating regime of the neurons. Activities and corresponding energies correspond to single trials, while the occupancy is averaged over the trajectories of $N_{ag} = 10$ agents. **(e)** Lifetime as a function of $\beta$. With reduced action magnitude, for small values of $\beta$, a small fraction of agents ($\sim 1$ out of the $N_{av} = 10$) fails to avoid the terminal state. **(f)** The exploitation of the non-saturating regime leads the NeuroMOP network to increase also $\langle \sigma_{\Delta X} \rangle$, i.e., the average standard deviation of the 'jumps' done by the activities in two consecutive temporal points in the dynamics. **(g)** Average standard deviation over the trajectories is not affected by $\beta$. **(h)** The effective dimensionality of the action signal is reduced for $\beta = 1.5$ in order to drive the dynamics in the 'rewarded' non saturating region. Averages are over $N_{av} = 10$ networks trained for $N_{ep} = 60$ epochs, with batches of $N_{traj} = 10$ trajectories. Standard deviations are computed over batches of $N_{traj} = 50$ trajectories. Errors are $SEM$.

the state space

$$\text{Vol}(dx') = \det(\hat{1} + \delta t \nabla f_\pi(x,a))\text{Vol}(dx) = \left(\hat{1} + \text{Tr}\left(\nabla f_\pi(x,a)\right)\right)\text{Vol}(dx) , \qquad (18)$$

where we exploited the fact that for small $\delta t$ the RNN dynamics behaves as a perturbation of an identity transformation and we can approximate the determinant with the trace. This 'extra' term $\text{Tr}(\nabla f_\pi(x,a))$ represents the contribution of the RNN dynamics in increasing the occupation of the state space. We take this term as the approximation of the state entropy $\mathcal{H}(\mathcal{S}|x,a)$ and add it to the intrinsic return the MOP agent receives along a trajectory. The value function is then modified as

$$V_\pi(x) = \mathrm{E}_{\tau \sim \pi, p}\left[\sum_{t=0}^{\infty}\gamma^t\left(\alpha\mathcal{H}\left(\mathcal{A}_t|x(t)\right) + \beta\text{Tr}(\nabla f_\pi(x(t), a(t)))\right)|x(0) = x\right] , \qquad (19)$$

where we introduced the 'temperature' hyperparameters $\alpha$ and $\beta$ regulating the amount of action and state entropy. Finding the optimal policy maximizing this value function reduces to solving the same constrained minimization problem defined in Eq. 13, with the value function $V_\pi(x)$ defined here. The resulting Lagrange multiplier follows the same expression as in Eq. 14, where the state entropy term $\mathcal{H}(\mathcal{S}|x,a)$ is substituted by our approximation $\text{Tr}(\nabla f_\pi(x,a))$. The optimal policy maximizing the value function defined above is

$$\pi^*(a|x) = \frac{1}{Z(x)}e^{\frac{1}{\alpha}\left(\beta\text{Tr}(\nabla f_\pi(x,a)) + \gamma\sum_{x'}p(x'|x,a)V^*(x')\right)} . \qquad (20)$$

Effectively, we will set $\alpha = 1$ and measure the state entropy temperature in units of $\alpha$.

We test the effect of introducing the state entropy term in the NeuroMOP network while satisfying the energy constraint, as defined in Appendix B. We start with an RNN showcasing chaotic dynamics, with large energy consumption, when left free to evolve with no external currents, i.e., with the

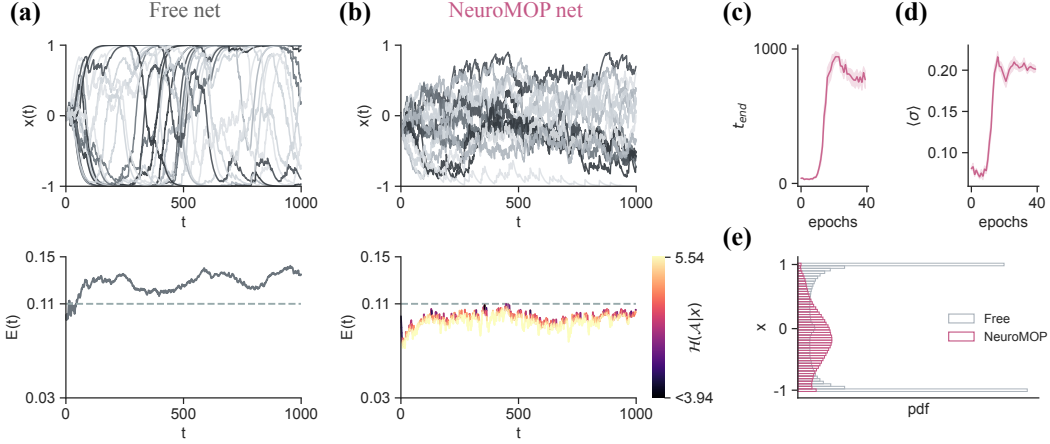

Figure 6: Energy constraint problem with noisy RNNs. **(a)** The RNN with no external control ($I = 0$) now has an intrinsic source of noise in the dynamics ($n = 0.1$, parameter scaling the external Gaussian noise term $\xi(t)$) (top panel). The energy is also subject to small fluctuations due to the noisy nature of the activities (bottom). **(b)** Given the external source of random noise, the NeuroMOP network learns to generate long lifetimes by keeping the energy close, but far enough from threshold. **(c)** The NeuroMOP network learns to avoid the terminal states and generate variability, but in this case the lifetime does not reach (on average) the maximum episode length. **(d)** On the other side, the average standard deviation of the trajectories increases with learning. **(e)** Average probability density function of the occupancy in the activity space. Averages over $N_{av} = 10$ networks with batches of $N_{traj} = 10$ trajectories, errors are $SEM$.

dynamics defined in Eq. 1 with $I = 0$ (Fig. 6a). Without the state-entropy term ($\beta = 0$), the choice of the parameter $\rho$ ($\rho = 1.2$) rescaling the external current is such that the injected external currents cannot lead to the full occupancy of the state space. Thus, the NeuroMOP network learns to avoid the terminal states but still largely exploits the saturating states of the transfer function (Fig. 6b). With increasing $\beta$, the NeuroMOP network gradually expands its occupation in the activity space, favoring those regions leading to larger activity changes (Fig. 6c,d). First, we note that we can match the average lifetimes for different values of $\beta$ (Fig. 6e). To test the effect of introducing the state entropy term, we define for each neuron $i$ the activity 'jumps' $\Delta x_i$ as the difference in activity between two consecutive time points, i.e., $\Delta x_i(t) = x_i(t+1) - x_i(t) \ \forall i = 1, \ldots, N$. Therefore, we introduce the standard deviation of these series averaged across neurons, i.e., $\langle \sigma_{\Delta x} \rangle$, and find that it increases with the contribution of the (approximated) state entropy (Fig. 6f). Conversely, the average standard deviation remains roughly constant (Fig. 6g). To drive neurons towards the highly sensitive region of the transfer function, the NeuroMOP network reduces its effective action dimensionality even when far from the threshold, deviating from the maximum available dimensionality when $\beta = 1.5$ (Fig. 6h).

## E   RNNs with noisy dynamics

We extend the NeuroMOP network to a controller of a noisy RNN, following the same dynamics as in Eq. 1 but with an additional Gaussian noise term as

$$x_i(t+1) = x_i(t) + \delta t \left( -\frac{x_i}{\tau} + \Phi \left( \sum_{j=1}^{N} J_{ij} x_j + I_i(t) + n \xi_i(t) \right) \right) , \qquad (21)$$

where $\xi_i \in \mathcal{N}(0, 1)$ is an i.i.d. normal random variable acting on each neuron $i$, and $n$ is the noise amplitude. With the introduction of noise in the network, we consider an environment whose transition probability is defined over a continuous set of possible values. Therefore, sampling from the optimal policy requires performing the integral that appears in the r.h.s of $\pi^*(a|x) = \frac{1}{Z(x)} e^{\gamma \int_{x'} p(x'|x,a) V(x')}$.

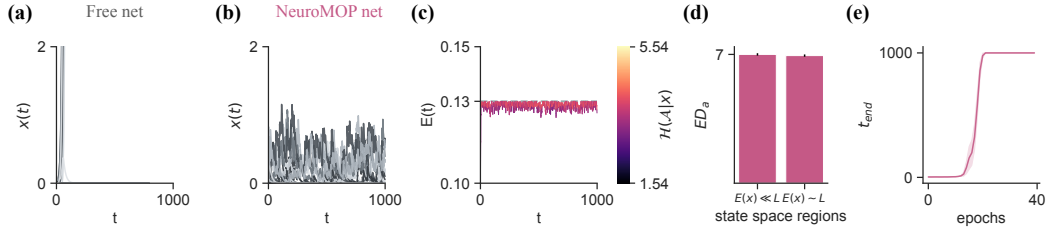

Figure 7: Energy constraint problem with non-saturating transfer function ($\Phi(\cdot) = \text{ReLU}(\cdot)$). **(a)** The free network (Eq. 1 with $I = 0$) is characterized by exploding patterns of the RNN's activity. Consequently, its energy consumption quickly diverges as well (not shown). **(b-c)** The NeuroMOP network learns to keep the RNN's activity bounded (b) by avoiding a terminal state of high energy consumption (c). Here, we choose an arbitrary threshold of $L = 0.13$. **(d)** Effective action dimensionality when far ($E(x) \ll L$) and close ($E(x) \sim L$) to the terminal state. **(e)** Average lifetime as a function of the training epochs. Averages over $N_{av} = 10$ networks with batches of $N_{traj} = 10$ trajectories, errors are $SEM$.

Here, we describe an approximation to the integral, suitable for our transition dynamics. For small $\delta t$, we can locally linearize the transfer function $\Phi(\cdot)$, and replace normal variable with a Bernoulli variable with $p = q = 0.5$. With this approximation, the integral simplifies to a sum of a positive and a negative contribution of the noise over the dynamics, namely

$$\int dx'\, p(x'|x, a) V(x') = q V(x'(x, a, \xi = 1)) + q V(x'(x, a, \xi = -1)), \qquad (22)$$

where $x'(x, a, \xi = \pm 1)$ defines the evolution of the dynamics as in Eq. 21 when the external noise takes the values $+1$ or $-1$, respectively.

The dynamics in Eq. 21 in the absence of external control ($I = 0$) generate noisy patterns of activation, with neurons, and consequently the energy, randomly fluctuating (Fig. 6a). We test the NeuroMOP network in the same energy constraint problem defined in Appendix B. Due to the inherent random fluctuations in the activities, long lifetimes are only granted if the RNN is kept sufficiently distant from the terminal state. The NeuroMOP network adopts this strategy, reducing its action entropy even before approaching proximity to the boundary (Fig. 6b). Nevertheless, the stochastic nature of the RNN can still lead the network to enter the terminal state after training: while the average lifetime approaches the arbitrary length of the simulation, it always remains below threshold (Fig. 6c). The NeuroMOP network learns to increase the average standard deviation of the individual trajectories (Fig. 6d), and to broadly occupy the activity space (Fig. 6e).

## F   RNNs with non-saturating transfer function

The ability of the NeuroMOP network to avoid terminal states and generate variability is independent on the choice of the RNN parameters and dynamics. To illustrate this point, we introduce RNNs following the dynamics defined in Eq. 1, but employing a non-saturating transfer function, specifically $\Phi(\cdot) = \text{ReLU}(\cdot)$. In the absence of external control (no external currents, $I = 0$), the RNN activity exhibits runaway excitation of its neurons (Fig. 7a), resulting in unbounded levels of energy consumption. We set thus an arbitrary threshold $L$ of high energy and test the NeuroMOP network's ability to inject variable currents while avoiding the terminal states $x^\dagger$ where $E(x^\dagger) > L$. The network successfully bounds the RNN's activity pattern (Fig. 7b) by keeping the energy below threshold for the whole duration of the episode (Fig. 7c). Notably, the inherent diverging drive of the network given by the linear transfer function, makes it highly responsive to positive input. To avoid this, the NeuroMOP network reduces action entropy throughout the whole episode, not only in proximity of the energy threshold. The energy constraint imposes in the action space a no-go direction for all states in state space due to the high susceptibility in the RNN. Thus, the effective action dimensionality loses one degree of freedom independently of the distance from the terminal state (Fig. 7d). The NeuroMOP network rapidly increase its lifetime through learning (Fig. 7e).

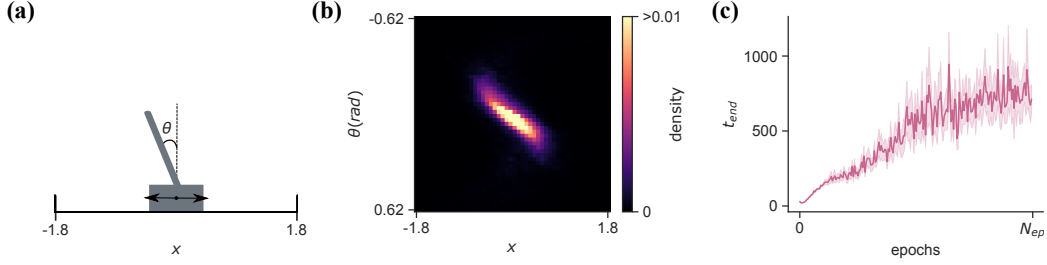

Figure 8: Balancing a cartpole. **(a)** Scheme of the cartpole. The controller network has binary actions (forces) $F \in \{-40, 40\}$ to act on the cart. **(b)** Probability density function of the occupation of the cart position $x$ and pole angle $\theta$. The MOP network balances the pole while generating variability in its state variables. **(c)** Lifetime increases with the training of the value function. Averages over $N_{av} = 10$ networks, errors are $SEM$.

Table 3: Hyperparameters for the balancing of the cartpole

| Parameter | Value |
|---|---|
| cart mass $M$ | 1.0 |
| pole mass $m$ | 0.1 |
| pole length $l$ | 1.0 |
| gravity acceleration $g$ | 9.81 |
| discount factor $\gamma$ | 0.98 |
| number of hidden layers (FFN) | 1 |
| hidden units per layer | 10 |
| input units $N_{inp}$ | 4 |
| FFN nonlinearity | ReLU |
| training epochs $N_{ep}$ | $10^5$ |
| number of agents $N_{ag}$ | 10 |
| trajectories per batch $N_{traj}$ | 20 |
| optimizer | SGD |
| learning rate $\eta$ | 0.02 |

## G   Balancing the Cartpole

We test the ability of MOP to control and generate diverse behavior in a system with physically realistic dynamics. We consider a MOP 'network' balancing a cartpole [69, 70] composed of a moving cart with a pole attached and free to rotate (Fig. 8a). The system has four degrees of freedom: the cart position $x$, the pole angle $\theta$ and the corresponding velocities, $\dot{x}$ and $\dot{\theta}$ respectively. The value function is approximated via a feedforward network receiving as input the four variables (hyperparameters are reported in Table 3). When not in a terminal state, the MOP network acts directly on the cart with two possible actions, which are binary forces $F \in \{-40, 40\}$, with dynamics

$$\ddot{\theta} = \frac{g \sin \theta + \cos(\theta) \left( \frac{-F - ml\dot{\theta}^2 + \sin \theta}{m + M} \right)}{l \left( \frac{4}{3} - \frac{m \cos^2 \theta}{m + M} \right)} \tag{23}$$

$$\ddot{x} = \frac{F + ml\dot{\theta}^2 \sin \theta - \ddot{\theta} \cos \theta}{M + m}, \tag{24}$$

where $M$ is the mass of the cart, $m$ and $l$ are the mass and the length of the pole and $g$ is the gravity acceleration. Note that no damping is applied to the cartpole, i.e., the system is frictionless. The MOP network enters a terminal state when either the cart position of the pole angle overshoot their threshold, specifically when $|x| > 1.8$ or $|\theta| > 0.62$ (radiant units). We threshold the values of the velocities such that they never exceeds the values $|\dot{x}| = 6$ and $|\dot{\theta}| = 3$. We find that the MOP network can generate variability in the cart position and the pole angle (Fig. 8b) while balancing the pole, ensuring enough distance to the terminal states (Fig. 8c).

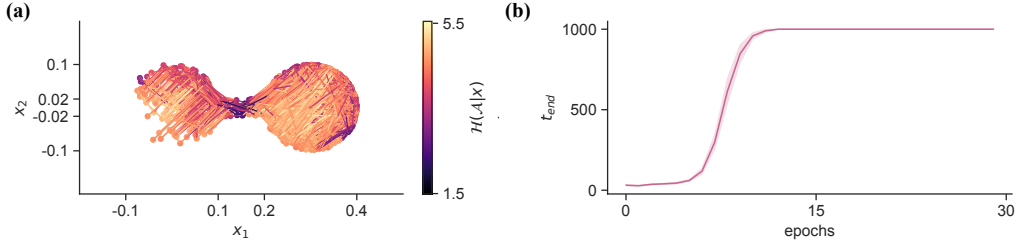

Figure 9: Two rooms arena connected by a narrow corridor in the neural space $(x_1, x_2)$. **(a)** The NeuroMOP network occupies both rooms of the arena and acts deterministically to cross the corridor (example of one trajectory with $T = 5000$), and **(b)** learns quickly to avoid the complex terminal states (note low action entropy at the corridor). Averages over $N_{av} = 10$ networks with batches of $N_{traj} = 10$ trajectories, errors are $SEM$.

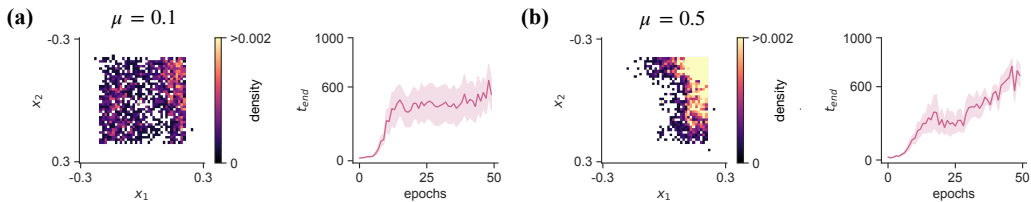

Figure 10: Adding an extrinsic reward term to MOP changes the behavior in the constrained neural space problem. Terminal states are all the states outside the boundaries of a square in the neural space $(x_1, x_2)$, centered in zero and with total side length $l = 0.4$. The NeuroMOP network gets an extrinsic reward $r = +1$ whenever in an inner square centered in $(0.1, 0.1)$ and with side length $l_r = 0.1$. We introduce a parameter $\mu$ regulating the balance between extrinsic reward and the action-entropy term. **(a)** Probability density function of the occupation (left) and lifetime (right) for $\mu = 0.1$. **(b)** Same as in (a), but for $\mu = 0.5$. Averages over $N_{av} = 5$ networks with batches of $N_{traj} = 10$ trajectories, errors are $SEM$.

## H   Crossing a narrow corridor

MOP agents' ability to generate future entropy is guaranteed by their capacity to flexibly switch between stochastic and deterministic behavior. We provide an additional example of this flexibility by showing the behavior of the NeuroMOP network in a problem where highly deterministic actions are (locally) required. Specifically, we consider the same neural space $(x_1, x_2)$ we introduced in Sec. 3.2, but now terminal states are defined such that the network can only live in a region of space defined by two circles (e.g., rooms) connected by a narrow available region of the neural space (e.g., a corridor). We observe that the intrinsic motivation for occupancy drives the NeuroMOP network to visit both rooms. To succeed in that, the NeuroMOP network largely reduces its action entropy to cross the narrow corridor, to later increase its stochasticity when in the larger rooms (Fig. 9a). Importantly, crossing the narrow neural space does not affect the agent's ability to avoid the terminal states (Fig. 9b).

## I   Adding an extrinsic reward

We showed that the NeuroMOP network is able to show complex behavior (e.g., crossing a narrow corridor) without the need to specify any reward function. Here we show that NeuroMOP is compatible with the addition of an extrinsic reward function. In this case, the value function is modified as

$$V_\pi(x) = \mathbb{E}_{\tau \sim \pi, p} \left[ \sum_{t=0}^{\infty} \gamma^t (\alpha \mathcal{H}(\mathcal{A}_t | x(t)) + \beta \mathcal{H}(\mathcal{S}_{t+1} | x(t), a(t)) + \mu r(x(t), a(t))) | x(0) = x \right],$$
(25)

where the hyperparameters $\alpha$, $\beta$ and $\mu$ regulate the weight on action entropy, state entropy and extrinsic reward maximization, respectively. By solving the same constrained minimization as in Eq. 13 we obtain the analogous formulation of the optimal policy maximizing the value function described above as

$$\pi^*(a|x) = \frac{1}{Z(x)} e^{\frac{1}{\alpha}\left(\mu r(x,a) + \beta \mathcal{H}(\mathcal{S}|x,a) + \gamma \sum_{x'} p(x'|x,a) V^*(x')\right)} \; . \tag{26}$$

As before, we set $\beta = 0$ and $\alpha = 1$ and measure the weight of the extrinsic reward in units of $\alpha$.

We have already shown that by setting specific terminal states structures we can have MOP agents generating behaviors without the need to specify any reward function. We test now the effect of providing the NeuroMOP network with an additional extrinsic reward. We consider the same problem as in Sec. 3.2 where the NeuroMOP network activity has to be confined in a square region of the neural space $(x_1, x_2)$, here centered in zero and of side length $l = 0.4$. Here, we modify the scenario by adding an additional state-dependent reward $r(x) = 1$ that the NeuroMOP network receives every time its activity $x$ is in a smaller square of the $(x_1, x_2)$ space, centered in $(0.1, 0.1)$ and with side length $l_r = 0.1$. By modulating the term $\mu$ regulating the importance of the extrinsic reward, we observe different behaviors. When $\mu$ is small enough ($\mu = 0.1$), the NeuroMOP network shows only a slight preference for the rewarded region of the square, maintaining an overall uniform occupancy (Fig. 10 a). In contrast, as $\mu$ increases ($\mu = 0.5$), behavior collapses to the occupation of the extrinsically rewarded square (Fig. 10 b). Finally, as the rewarded region is not in conflict with terminal states, a more conservative behavior is also associated with a decreased variability of the lifetime (Fig. 10). Our results highlight a significant distinction between MOP and the standard extrinsic reward maximization framework: the introduction of extrinsic rewards is associated to a collapse of the repertoire of the observed behavior into simpler behavioral patterns. Certainly, while the agent in both cases acts to maximize what is asked to, the critical question remains what constitutes interesting behavior, as we have argued throughout this paper.

## J Pseudocode

We show here the pseudocode of the algorithm (Pseudocode 1). While sampling from the policy, the network receives a teaching signal regarding the presence of terminal states. Note that we enforce that the value function approximation has value zero in all terminal states regardless of their approximated value. Also, to enhance stability, the contribution to the loss function coming from visited terminal states have been weighted by a factor one order of magnitude larger.

**Data:** Random $J$, $w$, $K$, $x_0$

**for** $i : 1$ **to** *N_ep* **do**

    Initialize $\mathcal{L} = 0$;

    **for** $traj : 1$ **to** *N_traj* **do**

        $x \leftarrow x_0$;

        **for** $t : 1$ **to** *T* **do**

            $V(x, w) \leftarrow x, w$;

            **if** $x > x_{th}$ **then**

                $\mathcal{L} + = 1. \left(V(x, w) - 0\right)^2$ ;

                $t_{end} \leftarrow t$ ;

                break

            **end**

            **else**

                $Z \leftarrow \sum_a e^{\gamma V(x'(x,a),w)}$;

                $V_{\mathcal{B}}(x, w) \leftarrow lnZ$;

                $\mathcal{L} + = 0.1 \left(V(x, w) - V_{\mathcal{B}}(x, w)\right)^2$ ;

            **end**

            $a \sim \pi(\cdot|x)$ ;

            $x \leftarrow \text{RNN}(x,a)$;

        **end**

    **end**

    $w \leftarrow \text{Adam}(w, L)$

**end**

    **Algorithm 1:** Pseudo-code for the training of the weights $w$ of the feedforward network.

